# Image representations for facial expression coding

**Marian Stewart Bartlett*** 
U.C. San Diego 
marni@salk.edu

**Gianluca Donato** 
Digital Persona, Redwood City, CA 
gianlucad@digitalpersona.com

**Javier R. Movellan** 
U.C. San Diego 
movellan@cogsci.ucsd.edu

**Joseph C. Hager** 
Network Information Res., SLC, Utah 
jchager@ibm.com

**Paul Ekman** 
U.C. San Francisco 
ekman@compuserve.com

**Terrence J. Sejnowski** 
Howard Hughes Medical Institute 
The Salk Institute; U.C. San Diego 
terry@salk.edu

## Abstract

The Facial Action Coding System (FACS) (9) is an objective method for quantifying facial movement in terms of component actions. This system is widely used in behavioral investigations of emotion, cognitive processes, and social interaction. The coding is presently performed by highly trained human experts. This paper explores and compares techniques for automatically recognizing facial actions in sequences of images. These methods include unsupervised learning techniques for finding basis images such as principal component analysis, independent component analysis and local feature analysis, and supervised learning techniques such as Fisher's linear discriminants. These data-driven bases are compared to Gabor wavelets, in which the basis images are predefined. Best performances were obtained using the Gabor wavelet representation and the independent component representation, both of which achieved 96% accuracy for classifying 12 facial actions. The ICA representation employs 2 orders of magnitude fewer basis images than the Gabor representation and takes 90% less CPU time to compute for new images. The results provide converging support for using local basis images, high spatial frequencies, and statistical independence for classifying facial actions.

## 1 Introduction

Facial expressions provide information not only about affective state, but also about cognitive activity, temperament and personality, truthfulness, and psychopathology. The Facial Action Coding System (FACS) (9) is the leading method for measuring facial movement in behavioral science. FACS is performed manually by highly trained human experts. A FACS coder decomposes a facial expression into component muscle movements (Figure 1). Ekman and Friesen described 46 distinct facial movements, and over 7000 distinct combinations of such movements have

* To whom correspondence should be addressed. (UCSD 0523, La Jolla, CA 92093.) This research was supported by NIH Grant No. 1F32 MH12417-01.

been observed in spontaneous behavior. An automated system would make facial expression measurement more widely accessible as a research tool for behavioral science and medicine. Such a system would also have application in human-computer interaction tools and low bandwidth facial animation coding.

A number of systems have appeared in the computer vision literature for classifying facial expressions into a few basic categories of emotion, such as happy, sad, or surprised. While such approaches are important, an objective and detailed measure of facial activity such as FACS is needed for basic research into facial behavior. In a system being developed concurrently for automatic facial action coding, Cohn and colleagues (7) employ feature point tracking of a select set of image points. Techniques employing 2-D image filters have proven to be more effective than feature-based representations for face image analysis [e.g. (6)]. Here we examine image analysis techniques that densely analyze graylevel information in the face image.

This work surveys and compares techniques for face image analysis as applied to automated FACS encoding.[1] The analysis focuses on methods for face image representation in which image graylevels are described as a linear superposition of basis images. The techniques were compared on a common image testbed using common similarity measures and classifiers.

We compared four representations in which the basis images were learned from the statistics of the face image ensemble. These include unsupervised learning techniques such as principal component analysis (PCA), and local feature analysis (LFA), which are learned from the second-order dependences among the image pixels, and independent component analysis (ICA) which is learned from the high-order dependencies as well. We also examined a representation obtained through supervised learning on the second-order image statistics, Fisher's linear discriminants (FLD). Classification performances with these data-driven basis images were compared to Gabor wavelets, in which the basis images were pre-defined. We examined properties of optimal basis images, where optimal was defined in terms of classification.

Generalization to novel faces was evaluated using leave-one-out cross-validation. Two basic classifiers were employed: nearest neighbor and template matching, where the templates were the mean feature vectors for each class. Two similarity measures were employed for each classifier: Euclidean distance and cosine of the angle between feature vectors.

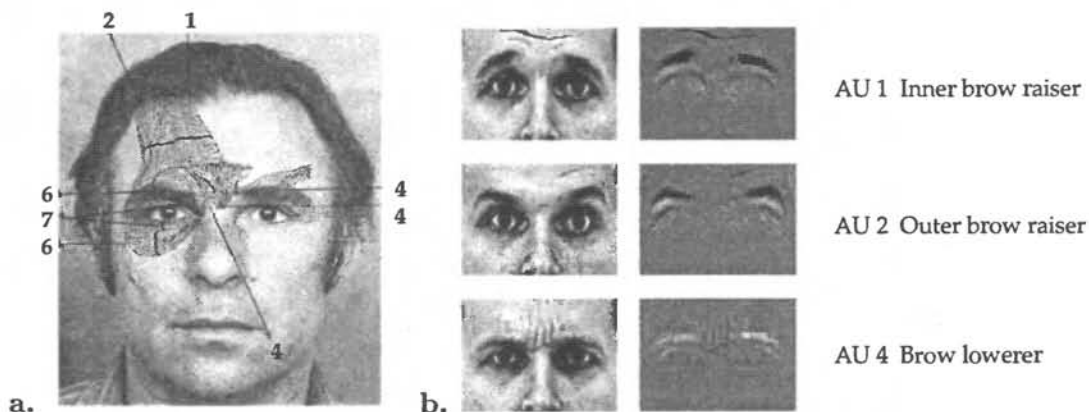

Figure 1: a. The facial muscles underlying six of the 46 facial actions. b. Cropped face images and $\delta$-images for three facial actions (AU's).

## 2   Image Database

We collected a database of image sequences of subjects performing specified facial actions. The database consisted of image sequences of subjects performing specified facial actions. Each sequence began with a neutral expression and ended with a high magnitude muscle contraction. For this investigation, we used 111 sequences from 20 subjects and attempted to classify 12 actions: 6 upper face actions and 6 lower face actions. Upper and lower-face actions were analyzed separately since facial motions in the lower face do not effect the upper face, and vice versa (9).

The face was located in the first frame in each sequence using the centers of the eyes and mouth. These coordinates were obtained manually by a mouse click. The coordinates from Frame 1 were used to register the subsequent frames in the sequence. The aspect ratios of the faces were warped so that the eye and mouth centers coincided across all images. The three coordinates were then used to rotate the eyes to horizontal, scale, and finally crop a window of $60 \times 90$ pixels containing the upper or lower face. To control for variations in lighting, logistic thresholding and luminance scaling was performed (13). Difference images ($\delta$-images) were obtained by subtracting the neutral expression in the first image of each sequence from the subsequent images in the sequence.

## 3   Unsupervised learning

### 3.1   Eigenfaces (PCA)

A number of approaches to face image analysis employ data-driven basis vectors learned from the statistics of the face image ensemble. Techniques such as eigenfaces (17) employ principal component analysis, which is an unsupervised learning method based on the second-order dependencies among the pixels (the pixelwise covariances). PCA has been applied successfully to recognizing facial identity (17), and full facial expressions (14).

Here we performed PCA on the dataset of $\delta$-images, where each $\delta$-image comprised a point in $R^n$ given by the brightness of the $n$ pixels. The PCA basis images were the eigenvectors of the covariance matrix (see Figure 2a), and the first $p$ components comprised the representation. Multiple ranges of components were tested, from $p = 10$ to $p = 200$, and performance was also tested excluding the first 1-3 components. Best performance of 79.3% correct was obtained with the first 30 principal components, using the Euclidean distance similarity measure and template matching classifier.

Padgett and Cottrell (14) found that a local PCA representation outperformed global PCA for classifying full facial expressions of emotion. Following the methods in (14), a set of local basis images was derived from the principal components of $15 \times 15$ image patches from randomly sampled locations in the $\delta$–images (see Figure 2d.) The first $p$ principal components comprised a basis set for all image locations, and the representation was downsampled by a factor of 4. Best performance of 73.4% was obtained with components 2-30, using Euclidean distance and template matching. Unlike the findings in (14), local basis images obtained through PCA were not more effective than global PCA for facial action coding. A second local implementation of PCA, in which the principal components were calculated for *fixed* $15 \times 15$ image patches also failed to improve over global PCA.

### 3.2   Local Feature Analysis (LFA)

Penev and Atick (15) recently developed a local, topographic representation based on second-order image statistics called local feature analysis (LFA). The kernels are derived from the principal component axes, and consist of a "whitening" step to equalize the variance of the PCA coefficients, followed by a rotation to pixel space.

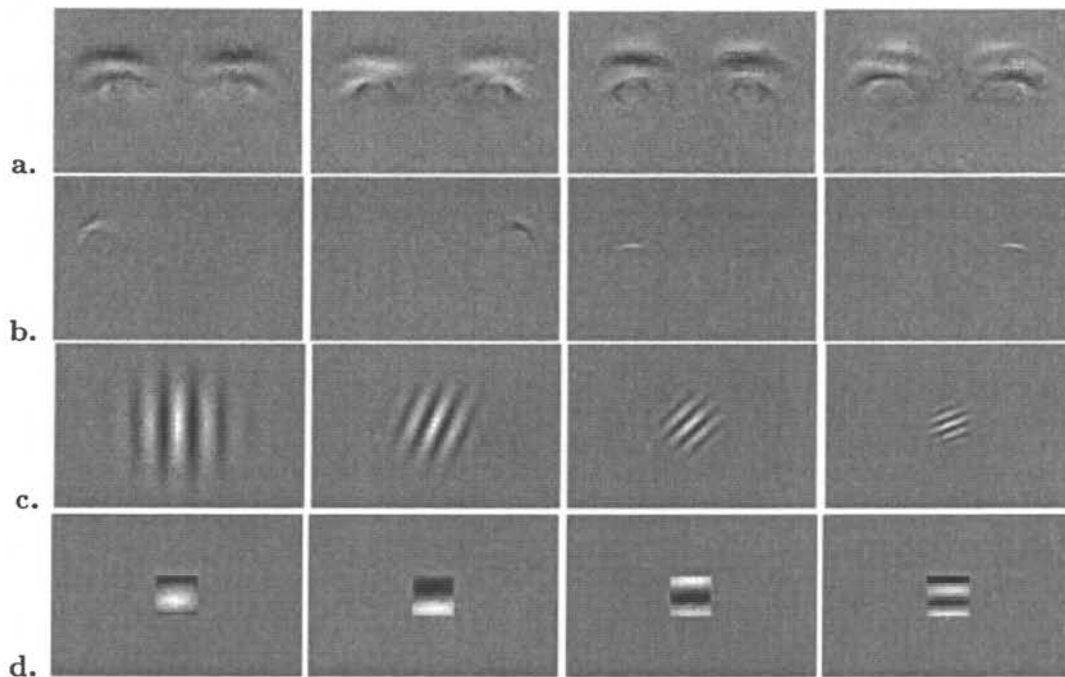

Figure 2: a. First 4 PCA basis images. b. Four ICA basis images. The ICA basis images are local, spatially opponent, and adaptive. c. Gabor kernels are local, spatially opponent, and predefined. d. First 4 local PCA basis images.

We begin with the matrix $P$ containing the principal component eigenvectors in its columns, and $\lambda_i$ are the corresponding eigenvalues. Each row of the matrix $K$ serves as an element of the LFA image dictionary[2]

$$K = PVP^T \quad \text{where} \quad V = D^{-\frac{1}{2}} = \mathrm{diag}(\frac{1}{\sqrt{\lambda_i}}) \quad i = 1,\ldots,p \qquad (1)$$

where $\lambda_i$ are the eigenvalues. The rows of $K$ were found to have spatially local properties, and are "topographic" in the sense that they are indexed by spatial location (15). The dimensionality of the LFA representation was reduced by employing an iterative sparsification algorithm based on multiple linear regression described in (15).

The LFA representation attained 81.1% correct classification performance. Best performance was obtained using the first 155 kernels, the cosine similarity measure, and nearest neighbor classifier. Classification performance using LFA was not significantly different from the performance using PCA. Although a face recognition algorithm based on the principles of LFA outperformed Eigenfaces in the March 1995 FERET competition, the exact algorithm has not been disclosed. Our results suggest that an aspect of the algorithm other than the LFA representation accounts for the difference in performance.

## 3.3 Independent Component Analysis (ICA)

Representations such as Eigenfaces, LFA, and FLD are based on the second-order dependencies among the pixels, but are insensitive to the high-order dependencies. High-order dependencies are relationships that cannot be described by a linear predictor. Independent component analysis (ICA) learns the high-order dependencies in addition to the second-order dependencies among the pixels.

The ICA representation was obtained by performing Bell & Sejnowski's infomax algorithm (4) (5) on the ensemble of $\delta$–images in the rows of the matrix $X$. The images in $X$ were assumed to be a linear mixture of an unknown set of independent source images which were recovered through ICA. In contrast to PCA, the ICA source images were local in nature (see Figure 2b). These source images provided a basis set for the expression images. The coefficients of each image with respect to the new basis set were obtained from the estimated mixing matrix $A \triangleq W^{-1}$, where $W$ is the ICA weight matrix [see (1), (2)].

Unlike PCA, there is no inherent ordering to the independent components of the dataset. We therefore selected as an ordering parameter the class discriminability of each component, defined as the ratio of between-class to within-class variance. Best performance of 95.5% was obtained with the first 75 components selected by class discriminability, using the cosine similarity measure, and nearest neighbor classifier. Independent component analysis gave the best performance among all of the data-driven image kernels. Class discriminability analysis of a PCA representation was previously found to have little effect on classification performance with PCA (2).

## 4    Supervised learning: Fisher's Linear Discriminants (FLD)

A class specific linear projection of a PCA representation of faces was recently shown to improve identity recognition performance (3). The method employs a classic pattern recognition technique, Fisher's linear discriminant (FLD), to project the images into a $c - 1$ dimensional subspace in which the $c$ classes are maximally separated. Best performance was obtained by choosing $p = 30$ principal components to first reduce the dimensionality of the data. The data was then projected down to 5 dimensions via the FLD projection matrix, $W_{fld}$. The FLD image dictionary was thus $W_{pca} * W_{fld}$. Best performance of 75.7% correct was obtained with the Euclidean distance similarity measure and template matching classifier.

FLD provided a much more compact representation than PCA. However, unlike the results obtained by (3) for identity recognition, Fisher's Linear Discriminants did not improve over basic PCA (Eigenfaces) for facial action classification. The difference in performance may be due to the low dimensionality of the final representation here. Class discriminations that are approximately linear in high dimensions may not be linear when projected down to as few as 5 dimensions.

## 5    Predefined image kernels: Gabor wavelets

An alternative to the adaptive bases described above are wavelet decompositions based on predefined families of Gabor kernels. Gabor kernels are 2-D sine waves modulated by a Gaussian (Figure 2c). Representations employing families of Gabor filters at multiple spatial scales, orientations, and spatial locations have proven successful for recognizing facial identity in images (11). Here, the $\delta$–images were convolved with a family of Gabor kernels $\psi_i$, defined as

$$\psi_i(\vec{x}) = \frac{\|\vec{k}_i\|^2}{\sigma^2} e^{-\frac{\|\vec{k}_i\|^2 \|\vec{x}\|^2}{2\sigma^2}} \left[ e^{j\vec{k}_i \vec{x}} - e^{-\frac{\sigma^2}{2}} \right] \tag{2}$$

where    $\vec{k}_i = \left( \begin{array}{c} f_\nu \cos\varphi_\mu \\ f_\nu \sin\varphi_\mu \end{array} \right)$,    $f_\nu = 2^{-\frac{\nu+2}{2}}\pi$,    $\varphi_\mu = \mu\frac{\pi}{8}$.

Following (11), the representation consisted of the amplitudes at 5 frequencies ($\nu = 0 - 4$) and 8 orientations ($\mu = 1 - 8$). Each filter output was downsampled by a factor $q$ and normalized to unit length. We tested the performance of the system using $q = 1, 4, 16$ and found that $q = 16$ yielded the best generalization rate. Best performance was obtained with the cosine similarity measure and nearest neighbor

classifier. Classification performance with the Gabor representation was 95.5%. This performance was significantly higher than all of the data-driven approaches in the comparison except independent component analysis, with which it tied.

## 6 Results and Conclusions

| PCA | Local PCA | LFA | ICA | FLD | Gabor |
|---|---|---|---|---|---|
| 79.3 ±3.9 | 73.4 ±4.2 | 81.1 ±3.7 | 95.5 ±2.0 | 75.7 ±4.1 | 95.5 ±2.0 |

Table 1: Summary of classification performance for 12 facial actions.

We have compared a number of different image analysis methods on a difficult classification problem, the classification of facial actions. Best performances were obtained with the Gabor and ICA representations, which both achieved 95.5% correct classification (see Table 1). The performance of these two methods equaled the agreement level of expert human subjects on these images (94%). Image representations derived from the second-order statistics of the dataset (PCA and LFA) performed in the 80% accuracy range. An image representation derived from supervised learning on the second-order statistics (FLD) also did not significantly differ from PCA. We also obtained evidence that high spatial frequencies are important for classifying facial actions. Classification with the three highest frequencies of the Gabor representation ($\nu = 0, 1, 2$, cycles/face = 15,18,21 cycles/face) was 93% compared to 84% with the three lowest frequencies ($\nu = 2, 3, 4$, cycles/face = 9,12,15).

The two representations that significantly outperformed the others, Gabor and Independent Components, employed local basis images, which supports recent findings that local basis images are important for face image analysis (14) (10) (12). The local property alone, however, does not account for the good performance of these two representations, as LFA performed no better than PCA on this classification task, nor did local implementations of PCA.

In addition to spatial locality, the ICA representation and the Gabor filter representation share the property of redundancy reduction, and have relationships to representations in the visual cortex. The response properties of primary visual cortical cells are closely modeled by a bank of Gabor kernels. Relationships have been demonstrated between Gabor kernels and independent component analysis. Bell & Sejnowski (5) found using ICA that the kernels that produced independent outputs from natural scenes were spatially local, oriented edge kernels, similar to a bank of Gabor kernels. It has also been shown that Gabor filter outputs of natural images are at least pairwise independent (16).

The Gabor wavelets and ICA each provide a way to represent face images as a linear superposition of basis functions. Gabor wavelets employ a set of pre-defined basis images, whereas ICA learns basis images that are adapted to the data ensemble. The Gabor wavelets are not specialized to the particular data ensemble, but would be advantageous when the amount of data is small. The ICA representation has the advantage of employing two orders of magnitude fewer basis images. This can be an advantage for classifiers that involve parameter estimation. In addition, the ICA representation takes 90% less CPU time than the Gabor representation to compute once the ICA weights are learned, which need only be done once.

In summary, this comparison provided converging support for using local basis images, high spatial frequencies, and statistical independence for classifying facial actions. Best performances were obtained with Gabor wavelet decomposition and independent component analysis. These two representations employ graylevel basis functions that share properties of spatial locality, independence, and have relationships to the response properties of visual cortical neurons.

An outstanding issue is whether our findings depend on the simple recognition engines we employed. Would a smarter recognition engine alter the relative per-

formances? Our preliminary investigations suggest that is not the case. Hidden Markov models (HMM's) were trained on the PCA, ICA and Gabor representations. The Gabor representation was reduced to 75 dimensions using PCA before training the HMM. The HMM improved classification performance with ICA to 96.3%, and it did not change the overall findings, as it gave similar percent improvements to the PCA and PCA-reduced Gabor representations over their nearest neighbor performances. The dimensionality reduction of the Gabor representation, however, caused its nearest neighbor performance to drop, and the performance with the HMM was 92.7%. The lower dimensionality of the ICA representation was an advantage when employing the HMM.

# 7  References

[1] M.S. Bartlett. *Face Image Analysis by Unsupervised Learning and Redundancy Reduction*. PhD thesis, University of California, San Diego, 1998.

[2] M.S. Bartlett, H.M. Lades, and T.J. Sejnowski. Independent component representations for face recognition. In T. Rogowitz, B. & Pappas, editor, *Proceedings of the SPIE Symposium on Electonic Imaging: Science and Technology; Human Vision and Electronic Imaging III*, volume 3299, pages 528–539, San Jose, CA, 1998. SPIE Press.

[3] P.N. Belhumeur, J.P. Hespanha, and D.J. Kriegman. Eigenfaces vs. fisherfaces: Recognition using class specific linear projection. *IEEE Transations on Pattern Analysis and Machine Intelligence*, 19(7):711–720, 1997.

[4] A.J. Bell and T.J. Sejnowski. An information-maximization approach to blind separation and blind deconvolution. *Neural Computation*, 7(6):1129–1159, 1995.

[5] A.J. Bell and T.J. Sejnowski. The independent components of natural scenes are edge filters. *Vision Research*, 37(23):3327–3338, 1997.

[6] R. Brunelli and T. Poggio. Face recognition: Features versus templates. *IEEE transactions on pattern analysis and machine intelligence*, 15(10):1042–1052, 1993.

[7] J.F. Cohn, A.J. Zlochower, J.J. Lien, Y-T Wu, and T. Kanade. Automated face coding: A computer-vision based method of facial expression analysis. *Psychophysiology*, 35(1):35–43, 1999.

[8] G. Donato, M. Bartlett, J. Hager, P. Ekman, and T. Sejnowski. Classifying facial actions. *IEEE Transactions on Pattern Analysis and Machine Intelligence*, 21(10):974–989, 1999.

[9] P. Ekman and W. Friesen. *Facial Action Coding System: A Technique for the Measurement of Facial Movement*. Consulting Psychologists Press, Palo Alto, CA, 1978.

[10] M.S. Gray, J. Movellan, and T.J. Sejnowski. A comparison of local versus global image decomposition for visual speechreading. In *Proceedings of the 4th Joint Symposium on Neural Computation*, pages 92–98. Institute for Neural Computation, La Jolla, CA, 92093-0523, 1997.

[11] M. Lades, J. Vorbrüggen, J. Buhmann, J. Lange, W. Konen, C. von der Malsburg, and R. Würtz. Distortion invariant object recognition in the dynamic link architecture. *IEEE Transactions on Computers*, 42(3):300–311, 1993.

[12] D.D. Lee and S. Seung. Learning the parts of objects by non-negative matrix factorization. *Nature*, 401:788–791, 1999.

[13] J.R. Movellan. Visual speech recognition with stochastic networks. In G. Tesauro, D.S. Touretzky, and T. Leen, editors, *Advances in Neural Information Processing Systems*, volume 7, pages 851–858. MIT Press, Cambridge, MA, 1995.

[14] C. Padgett and G. Cottrell. Representing face images for emotion classification. In M. Mozer, M. Jordan, and T. Petsche, editors, *Advances in Neural Information Processing Systems*, volume 9, Cambridge, MA, 1997. MIT Press.

[15] P.S. Penev and J.J. Atick. Local feature analysis: a general statistical theory for object representation. *Network: Computation in Neural Systems*, 7(3):477–500, 1996.

[16] E. P. Simoncelli. Statistical models for images: Compression, restoration and synthesis. In *31st Asilomar Conference on Signals, Systems and Computers*, Pacific Grove, CA, November 2-5 1997.

[17] M. Turk and A. Pentland. Eigenfaces for recognition. *Journal of Cognitive Neuroscience*, 3(1):71–86, 1991.

## Footnotes

[1]A detailed description of this work appears in (8).

[2]An image dictionary is a set of images that decomposes other images, e.g. through inner product. Here it finds the coefficients for the basis set $K^{-1}$
